# Support Vector Machines with a Reject Option

**Yves Grandvalet** [1,2], **Alain Rakotomamonjy** [3], **Joseph Keshet** [2] and **Stéphane Canu** [3]

| [1] Heudiasyc, UMR CNRS 6599 | [2] Idiap Research Institute |
|:---:|:---:|
| Université de Technologie de Compiègne | Centre du Parc |
| BP 20529, 60205 Compiègne Cedex, France | CP 592, CH-1920 Martigny Switzerland |

[3] LITIS, EA 4108
Université de Rouen & INSA de Rouen
76801 Saint Etienne du Rouvray, France

## Abstract

We consider the problem of binary classification where the classifier may abstain instead of classifying each observation. The Bayes decision rule for this setup, known as Chow's rule, is defined by two thresholds on posterior probabilities. From simple desiderata, namely the consistency and the sparsity of the classifier, we derive the double hinge loss function that focuses on estimating conditional probabilities only in the vicinity of the threshold points of the optimal decision rule. We show that, for suitable kernel machines, our approach is universally consistent. We cast the problem of minimizing the double hinge loss as a quadratic program akin to the standard SVM optimization problem and propose an active set method to solve it efficiently. We finally provide preliminary experimental results illustrating the interest of our constructive approach to devising loss functions.

## 1 Introduction

In decision problems where errors incur a severe loss, one may have to build classifiers that abstain from classifying ambiguous examples. Rejecting these examples has been investigated since the early days of pattern recognition. In particular, Chow (1970) analyses how the error rate may be decreased thanks to the reject option.

There have been several attempts to integrate a reject option in Support Vector Machines (SVMs), using strategies based on the thresholding of SVMs scores (Kwok, 1999) or on a new training criterion (Fumera & Roli, 2002). These approaches have however critical drawbacks: the former is not consistent and the latter leads to considerable computational overheads to the original SVM algorithm and lacks some of its most appealing features like convexity and sparsity.

We introduce a piecewise linear and convex training criterion dedicated to the problem of classification with the reject option. Our proposal, inspired by the probabilistic interpretation of SVM fitting (Grandvalet et al., 2006), is a double hinge loss, reflecting the two thresholds in Chow's rule. Hence, we generalize the loss suggested by Bartlett and Wegkamp (2008) to arbitrary asymmetric misclassification and rejection costs. For the symmetric case, our probabilistic viewpoint motivates another decision rule. We then propose the first algorithm specifically dedicated to train SVMs with a double hinge loss. Its implementation shows that our decision rule is at least at par with the one of Bartlett and Wegkamp (2008).

The paper is organized as follows. Section 2 defines the problem and recalls Bayes rule for binary classification with a reject option. The proposed double hinge loss is derived in Section 3, together with the decision rule associated with SVM scores. Section 4 addresses implementation issues: it formalizes the SVM training problem and details an active set algorithm specifically designed for

training with the double hinge loss. This implementation is tested empirically in Section 5. Finally, Section 6 concludes the paper.

## 2 Problem Setting and the Bayes Classifier

Classification aims at predicting a class label $y \in \mathcal{Y}$ from an observed pattern $\mathbf{x} \in \mathcal{X}$. For this purpose, we construct a decision rule $d : \mathcal{X} \to \mathcal{A}$, where $\mathcal{A}$ is a set of actions that typically consists in assigning a label to $\mathbf{x} \in \mathcal{X}$. In binary problems, where the class is tagged either as $+1$ or $-1$, the two types of errors are: (i) false positive, where an example labeled $-1$ is predicted as $+1$, incurring a cost $c_-$; (ii) false negative, where an example labeled $+1$ is predicted as $-1$, incurring a cost $c_+$.

In general, the goal of classification is to predict the true label for an observed pattern. However, patterns close to the decision boundary are misclassified with high probability. This problem becomes especially eminent in cases where the costs, $c_-$ or $c_+$, are high, such as in medical decision making. In these processes, it might be better to alert the user and abstain from prediction. This motivates the introduction of a *reject option* for classifiers that cannot predict a pattern with enough confidence. This decision to abstain, which is denoted by $0$, incurs a cost, $r_-$ and $r_+$ for examples labeled $-1$ and $+1$, respectively.

The costs pertaining to each possible decision are recapped on the right-hand-side. In what follows, we assume that all costs are strictly positive:

$$c_- > 0 \;,\; c_+ > 0 \;,\; r_- > 0 \;,\; r_+ > 0 \quad . \tag{1}$$

Furthermore, it should be possible to incur a lower expected loss by choosing the reject option instead of any prediction, that is

$$c_- \, r_+ + c_+ \, r_- < c_- \, c_+ \quad . \tag{2}$$

|          |      | $y$   |       |
|----------|------|-------|-------|
|          |      | $+1$  | $-1$  |
|          | $+1$ | $0$   | $c_-$ |
| $d(\mathbf{x})$ | $0$  | $r_+$ | $r_-$ |
|          | $-1$ | $c_+$ | $0$   |

Bayes' decision theory is the paramount framework in statistical decision theory, where decisions are taken to minimize expected losses. For classification with a reject option, the overall risk is

$$
\begin{aligned}
R(d) \;=\;& c_+ \, \mathrm{E}_{XY}\left[ Y = 1, \, d(X) = -1 \right] + c_- \, \mathrm{E}_{XY}\left[ Y = -1, \, d(X) = 1 \right] + \\
& r_+ \, \mathrm{E}_{XY}\left[ Y = 1, \, d(X) = 0 \right] + r_- \, \mathrm{E}_{XY}\left[ Y = -1, \, d(X) = 0 \right] \;,
\end{aligned}
\tag{3}
$$

where $X$ and $Y$ denote the random variable describing patterns and labels.

The Bayes classifier $d^*$ is defined as the minimizer of the risk $R(d)$. Since the seminal paper of Chow (1970), this rule is sometimes referred to as *Chow's rule*:

$$
d^*(\mathbf{x}) = \begin{cases} +1 & \text{if } \mathrm{P}(Y = 1 | X = \mathbf{x}) > p_+ \\ -1 & \text{if } \mathrm{P}(Y = 1 | X = \mathbf{x}) < p_- \\ 0 & \text{otherwise} \;, \end{cases}
\tag{4}
$$

$$\text{where } \; p_+ = \frac{c_- - r_-}{c_- - r_- + r_+} \;\; \text{and} \;\; p_- = \frac{r_-}{c_+ - r_+ + r_-} \quad .$$

Note that, assuming that (1) and (2) hold, we have $0 < p_- < p_+ < 1$.

One of the major inductive principle is the empirical risk minimization, where one minimizes the empirical counterpart of the risk (3). In classification, this principle usually leads to a NP-hard problem, which can be circumvented by using a smooth proxy of the misclassification loss. For example, Vapnik (1995) motivated the hinge loss as a "computationally simple" (i.e., convex) surrogate of classification error. The following section is dedicated to the construction of such a surrogate for classification with a reject option.

## 3 Training Criterion

One method to get around the hardness of learning decision functions is to replace the conditional probability $\mathrm{P}(Y = 1 | X = \mathbf{x})$ with its estimation $\widehat{\mathrm{P}}(Y = 1 | X = \mathbf{x})$, and then plug this estimation back in (4) to build a classification rule (Herbei & Wegkamp, 2006). One of the most widespread

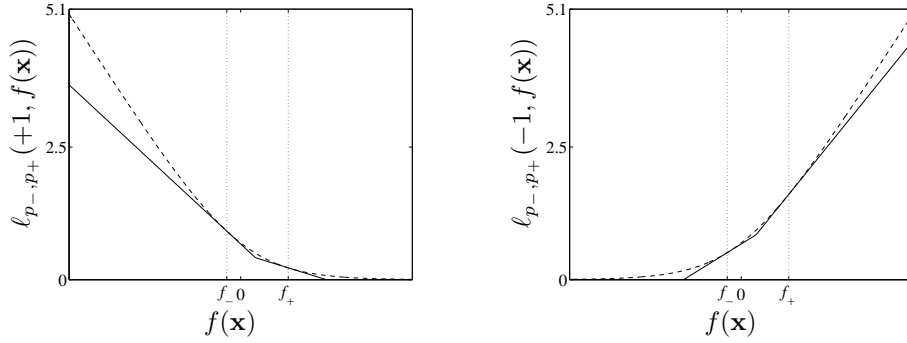

Figure 1: Double hinge loss function $\ell_{p_-,p_+}$ for positive (left) and negative examples (right), with $p_- = 0.4$ and $p_+ = 0.8$ (solid: double hinge, dashed: likelihood). Note that the decision thresholds $f_+$ and $f_-$ are not symmetric around zero.

representative of this line of attack is the logistic regression model, which estimates the conditional probability using the maximum (penalized) likelihood framework.

As a starting point, we consider the generalized logistic regression model for binary classification, where

$$\widehat{P}(Y = y | X = \mathbf{x}) = \frac{1}{1 + \exp(-yf(\mathbf{x}))} \ , \tag{5}$$

and the function $f : \mathcal{X} \to \mathbb{R}$ is estimated by the minimization of a regularized empirical risk on the training sample $\mathcal{T} = \{(\mathbf{x}_i, y_i)\}_{i=1}^n$

$$\sum_{i=1}^n \ell(y_i, f(\mathbf{x}_i)) + \lambda \Omega(f) \ , \tag{6}$$

where $\ell$ is a loss function and $\Omega(\cdot)$ is a regularization functional, such as the (squared) norm of $f$ in a suitable Hilbert space $\Omega(f) = \|f\|_{\mathcal{H}}^2$, and $\lambda$ is a regularization parameter. In the standard logistic regression procedure, $\ell$ is the negative log-likelihoood loss

$$\ell(y, f(\mathbf{x})) = \log(1 + \exp(-yf(\mathbf{x}))) \ .$$

This loss function is convex and decision-calibrated (Bartlett & Tewari, 2007), but it lacks an appealing feature of the hinge loss used in SVMs, that is, it does not lead to sparse solutions. This drawback is the price to pay for the ability to estimate the posterior probability $P(Y = 1 | X = \mathbf{x})$ on the whole range $(0, 1)$ (Bartlett & Tewari, 2007).

However, the definition of the Bayes' rule (4) clearly shows that the estimation of $P(Y = 1 | X = \mathbf{x})$ does not have to be accurate everywhere, but only in the vicinity of $p_+$ and $p_-$. This motivates the construction of a training criterion that focuses on this goal, without estimating $P(Y = 1 | X = \mathbf{x})$ on the whole range as an intermediate step. Our purpose is to derive such a loss function, without sacrifying sparsity to the consistency of the decision rule.

Though not a proper negative log-likelihood, the hinge loss can be interpreted in a maximum a posteriori framework: The hinge loss can be derived as a relaxed minimization of negative log-likelihood (Grandvalet et al., 2006). According to this viewpoint, minimizing the hinge loss aims at deriving a loose approximation to the the logistic regression model (5) that is accurate only at $f(\mathbf{x}) = 0$, thus allowing to estimate whether $P(Y = 1 | X = \mathbf{x}) > 1/2$ or not. More generally, one can show that, in order to have a precise estimate of $P(Y = 1 | X = \mathbf{x}) = p$, the surrogate loss should be tangent to the neg-log-likelihood at $f = \log(p/(1 - p))$.

Following this simple constructive principle, we derive the *double hinge loss*, which aims at reliably estimating $P(Y = 1 | X = \mathbf{x})$ at the threshold points $p_+$ and $p_-$. Furthermore, to encourage sparsity, we set the loss to zero for all points classified with high confidence. This loss function is displayed in Figure 1. Formally, for the positive examples, the double hinge loss satisfying the above conditions can be expressed as

$$\ell_{p_-,p_+}(+1, f(\mathbf{x})) = \max\left\{-(1 - p_-)f(\mathbf{x}) + H(p_-), -(1 - p_+)f(\mathbf{x}) + H(p_+), 0\right\} \ , \tag{7}$$

and for the negative examples it can be expressed as

$$\ell_{p_-,p_+}(-1, f(\mathbf{x})) = \max \big\{ p_+ f(\mathbf{x}) + H(p_+),\ p_- f(\mathbf{x}) + H(p_-),\ 0 \big\} , \qquad (8)$$

where $H(p) = -p\log(p) - (1-p)\log(1-p)$. Note that, unless $p_- = 1 - p_+$, there is no simple symmetry with respect to the labels.

After training, the decision rule is defined as the plug-in estimation of (4) using the logistic regression probability estimation. Let $f_+ = \log(p_+/(1-p_+))$ and $f_- = \log(p_-/(1-p_-))$, the decision rule can be expressed in terms of the function $f$ as follows

$$d_{p_-,p_+}(\mathbf{x}; f) = \begin{cases} +1 & \text{if } f(\mathbf{x}) > f_+ \\ -1 & \text{if } f(\mathbf{x}) < f_- \\ 0 & \text{otherwise .} \end{cases} \qquad (9)$$

The following result shows that the rule $d_{p_-,p_+}(\cdot; f)$ is *universally consistent* when $f$ is learned by minimizing empirical risk based on $\ell_{p_-,p_+}$. Hence, in the limit, learning with the double hinge loss is optimal in the sense that the risk for the learned decision rule converges to the Bayes' risk.

**Theorem 1.** *Let $\mathcal{H}$ be a functional space that is dense in the set of continuous functions. Suppose that we have a positive sequence $\{\lambda_n\}$ with $\lambda_n \to 0$ and $n\lambda_n^2/\log n \to \infty$. We define $f_n^*$ as*

$$\arg\min_{f \in \mathcal{H}} \frac{1}{n} \sum_{i=1}^{n} \ell_{p_-,p_+}(y_i, f(\mathbf{x}_i)) + \lambda_n \|f\|_{\mathcal{H}}^2 .$$

*Then,* $\lim_{n\to\infty} R(d_{p_-,p_+}(X; f_n^*)) = R(d^*(X))$ *holds almost surely, that is, the classifier* $d_{p_-,p_+}(\cdot; f_n^*)$ *is strongly universally consistent.*

*Proof.* Our theorem follows directly from (Steinwart, 2005, Corollary 3.15), since $\ell_{p_-,p_+}$ is *regular* (Steinwart, 2005, Definition 3.9). Besides mild regularity conditions that hold for $\ell_{p_-,p_+}$, a loss function is said regular if, for every $\alpha \in [0,1]$, and every $t_\alpha$ such that

$$t_\alpha = \arg\min_t \ \alpha\,\ell_{p_-,p_+}(+1, t) + (1-\alpha)\,\ell_{p_-,p_+}(-1, t) ,$$

we have that $d_{p_-,p_+}(t_\alpha, \mathbf{x})$ agrees with $d^*(\mathbf{x})$ almost everywhere.

Let $f_1 = -H(p_-)/p_-$, $f_2 = -(H(p_+) - H(p_-))/(p_+ - p_-)$ and $f_3 = H(p_+)/(1-p_+)$ denote the hinge locations in $\ell_{p_-,p_+}(\pm 1, f(\mathbf{x}))$. Note that we have $f_1 < f_- < f_2 < f_+ < f_3$, and that

$$t_\alpha \in \begin{cases} (-\infty, f_1] & \text{if } 0 \le \alpha < p_- \\ [f_1, f_2] & \text{if } \alpha = p_- \\ \{f_2\} & \text{if } p_- < \alpha < p_+ \\ [f_2, f_3] & \text{if } \alpha = p_+ \\ [f_3, \infty) & \text{if } p_+ < \alpha \le 1 \end{cases} \Rightarrow d_{p_-,p_+}(t_\alpha, \mathbf{x}) = \begin{cases} -1 & \text{if } \mathrm{P}(Y=1|\mathbf{x}) < p_- \\ -1 \text{ or } 0 & \text{if } \mathrm{P}(Y=1|\mathbf{x}) = p_- \\ 0 & \text{if } p_- < \mathrm{P}(Y=1|\mathbf{x}) < p_+ \\ 0 \text{ or } +1 & \text{if } \mathrm{P}(Y=1|\mathbf{x}) = p_+ \\ +1 & \text{if } \mathrm{P}(Y=1|\mathbf{x}) > p_+ \end{cases}$$

which is the desired result. $\square$

Note also that the analysis of Bartlett and Tewari (2007) can be used to show that minimizing $\ell_{p_-,p_+}$ cannot provide consistent estimates of $\mathrm{P}(Y=1|X=\mathbf{x}) = p$ for $p \notin \{p_-, p_+\}$. This property is desirable regarding sparsity, since sparseness does not occur when the conditional probabilities can be unambiguously estimated .

**Note on a Close Relative** A double hinge loss function has been proposed recently with a different perspective by Bartlett and Wegkamp (2008). Their formulation is restricted to symmetric classification, where $c_+ = c_- = 1$ and $r_+ = r_- = r$. In this situation, rejection may occur only if $0 \le r < 1/2$, and the thresholds on the conditional probabilities in Bayes' rule (4) are $p_- = 1 - p_+ = r$.

For symmetric classification, the loss function of Bartlett and Wegkamp (2008) is a scaled version of our proposal that leads to equivalent solutions for $f$, but our decision rule differs. While our probabilistic derivation of the double hinge loss motivates the decision function (9), the decision rule of Bartlett and Wegkamp (2008) has a free parameter (corresponding to the threshold $f_+ = -f_-$) whose value is set by optimizing a generalization bound.

Our decision rule rejects more examples when the loss incurred by rejection is small and fewer examples otherwise. The two rules are identical for $r \simeq 0.24$. We will see in Section 5 that this difference has noticeable outcomes.

# 4 SVMs with Double Hinge

In this section, we show how the standard SVM optimization problem is modified when the hinge loss is replaced by the double hinge loss. The optimization problem is first written using a compact notation, and the dual problem is then derived.

## 4.1 Optimization Problem

Minimizing the regularized empirical risk (6) with the double hinge loss (7–8) is an optimization problem akin to the standard SVM problem. Let $C$ be an arbitrary constant, we define $D = C(p_+ - p_-)$, $C_i = C(1 - p_+)$ for positive examples, and $C_i = Cp_-$ for negative examples. With the introduction of slack variables $\boldsymbol{\xi}$ and $\boldsymbol{\eta}$, the optimization problem can be stated as

$$\begin{cases} \min_{f,b,\boldsymbol{\xi},\boldsymbol{\eta}} & \frac{1}{2}\|f\|_{\mathcal{H}}^2 + \sum_{i=1}^{n} C_i \xi_i + D \sum_{i=1}^{n} \eta_i \\ \text{s.t.} & y_i(f(\mathbf{x}_i) + b) \geq t_i - \xi_i \quad i = 1, \ldots, n \\ & y_i(f(\mathbf{x}_i) + b) \geq \tau_i - \eta_i \quad i = 1, \ldots, n \\ & \xi_i \geq 0, \quad \eta_i \geq 0 \qquad\quad i = 1, \ldots, n, \end{cases} \tag{10}$$

where, for positive examples, $t_i = H(p_+)/(1 - p_+)$, $\tau_i = -(H(p_-) - H(p_+))/(p_- - p_+)$, while, for negative examples $t_i = H(p_-)/p_-$, $\tau_i = (H(p_-) - H(p_+))/(p_- - p_+)$.

For functions $f$ belonging to a Hilbert space $\mathcal{H}$ endowed with a reproducing kernel $k(\cdot, \cdot)$, efficient optimization algorithms can be drawn from the dual formulation:

$$\begin{cases} \min_{\boldsymbol{\alpha},\boldsymbol{\gamma}} & \frac{1}{2}\boldsymbol{\gamma}^T G \boldsymbol{\gamma} - \boldsymbol{\tau}^T \boldsymbol{\gamma} - (\mathbf{t} - \boldsymbol{\tau})^T \boldsymbol{\alpha} \\ \text{s.t.} & \mathbf{y}^T \boldsymbol{\gamma} = 0 \\ & 0 \leq \alpha_i \leq C_i \qquad\quad i = 1, \ldots, n \\ & 0 \leq \gamma_i - \alpha_i \leq D \quad i = 1, \ldots, n. \end{cases} \tag{11}$$

where $\mathbf{y} = (y_1, \ldots, y_n)^T$, $\mathbf{t} = (t_1, \ldots, t_n)^T$ and $\boldsymbol{\tau} = (\tau_1, \ldots, \tau_n)^T$ are vectors of $\mathbb{R}^n$ and $G$ is the $n \times n$ Gram matrix with general entry $G_{ij} = y_i y_j k(\mathbf{x}_i, \mathbf{x}_j)$. Note that (11) is a simple quadratic problem under box constraints. Compared to the standard SVM dual problem, one has an additional vector to optimize, but, with the active set we developed, we only have to optimize a single vector of $\mathbb{R}^n$. The primal variables $f$ and $b$ are then derived from the Karush-Kuhn-Tucker (KKT) conditions. For $f$, we have: $f(\cdot) = \sum_{i=1}^{n} \gamma_i y_i k(\cdot, \mathbf{x}_i)$, and $b$ is obtained in the optimization process described below.

## 4.2 Solving the Problem

To solve (11), we use an active set algorithm, following a strategy that proved to be efficient in SimpleSVM (Vishwanathan et al., 2003). This algorithm solves the SVM training problem by a greedy approach, in which one solves a series of small problems. First, the repartition of training examples in support and non-support vectors is assumed to be known, and the training criterion is optimized considering that this partition fixed. Then, this optimization results in an updated partition of examples in support and non-support vectors. These two steps are iterated until some level of accuracy is reached.

**Partitioning the Training Set** The training set is partitioned into five subsets defined by the activity of the box constraints of Problem (11). The training examples indexed by:

$I_0$ , defined by $I_0 = \{i | \gamma_i = 0\}$, are such that $y_i(f(\mathbf{x}_i) + b) > t_i$;

$I_t$ , defined by $I_t = \{i | 0 < \gamma_i < C_i\}$, are such that $y_i(f(\mathbf{x}_i) + b) = t_i$;

$I_C$ , defined by $I_C = \{i | \gamma_i = C_i\}$, are such that $\tau_i < y_i(f(\mathbf{x}_i) + b) \leq t_i$;

$I_\tau$ , defined by $I_\tau = \{i | C_i < \gamma_i < C_i + D\}$, are such that $y_i(f(\mathbf{x}_i) + b) = \tau_i$;

$I_D$ , defined by $I_D = \{i | \gamma_i = C_i + D\}$, are such that $y_i(f(\mathbf{x}_i) + b) \leq \tau_i$.

When example $i$ belongs to one of the subsets described above, the KKT conditions yield that $\alpha_i$ is either equal to $\gamma_i$ or constant. Hence, provided that the repartition of examples in the subsets $I_0$, $I_t$, $I_C$, $I_\tau$ and $I_D$ is known, we only have to consider a problem in $\boldsymbol{\gamma}$. Furthermore, $\gamma_i$ has to be computed only for $i \in I_t \cup I_\tau$.

**Updating Dual Variables**  Assuming a correct partition, Problem (11) reduces to the considerably smaller problem of computing $\gamma_i$ for $i \in I_T = I_t \cup I_\tau$:

$$
\begin{cases}
\min\limits_{\{\gamma_i | i \in I_T\}} & \dfrac{1}{2} \sum\limits_{i \in I_T, j \in I_T} \gamma_i \gamma_j G_{ij} - \sum\limits_{i \in I_T} \gamma_i s_i \\
\text{s. t.} & \sum\limits_{i \in I_T} y_i \gamma_i + \sum\limits_{i \in I_C} C_i y_i + \sum\limits_{i \in I_D} (C_i + D)\, y_i = 0 ~,
\end{cases}
\tag{12}
$$

where $s_i = t_i - \sum_{j \in I_C} C_j G_{ji} - \sum_{j \in I_D} (C_j + D)\, G_{ji}$ for $i \in I_t$ and $s_i = \tau_i - \sum_{j \in I_C} C_j G_{ji} - \sum_{j \in I_D} (C_j + D)\, G_{ji}$ for $i \in I_\tau$. Note that the box constraints of Problem (11) do not appear here, because we assumed the partition to be correct.

The solution of Problem (12) is simply obtained by solving the following linear system resulting from the first-order optimality conditions:

$$
\begin{cases}
\sum\limits_{j \in I_T} G_{ij} \gamma_j + y_i \lambda = s_i & \text{for } i \in I_T \\
\sum\limits_{i \in I_T} y_i \gamma_i = - \sum\limits_{i \in I_C} C_i y_i - \sum\limits_{i \in I_D} (C_i + D)\, y_i ~,
\end{cases}
\tag{13}
$$

where $\lambda$, which is the (unknown) Lagrange parameter associated to the equality constraint in (12), is computed along with $\boldsymbol{\gamma}$. Note that the $|I_T|$ equations of the linear system given on the first line of (13) express that, for $i \in I_t$, $y_i(f(\mathbf{x}_i) + \lambda) = t_i$ and for $i \in I_\tau$, $y_i(f(\mathbf{x}_i) + \lambda) = \tau_i$. Hence, the primal variable $b$ is equal to $\lambda$.

**Algorithm**  The algorithm, described in Algorithm 1, simply alternates updates of the partition of examples in $\{I_0, I_t, I_C, I_\tau, I_D\}$, and the ones of coefficients $\gamma_i$ for the current active set $I_T$. As for standard SVMs, the initialization step consists in either using the solution obtained for a different hyper-parameter, such as a higher value of $C$, or in picking one or several examples of each class to arbitrarily initialize $I_t$ to a non-empty set, and putting all the other ones in $I_0 = \{1, \ldots, n\} \setminus I_t$.

---

**Algorithm 1** SVM Training with a Reject Option

---

**input**  $\{\mathbf{x}_i, y_i\}_{1 \le i \le n}$ and hyper-parameters $C$, $p_+$, $p_-$
   initialize $\gamma^{\text{old}}$ $I_T = \{I_t, I_\tau\}$, $\overline{I_T} = \{I_0, I_C, I_D\}$,
  **repeat**
     solve linear system (13) $\rightarrow (\gamma_i)_{i \in I_T}, b = \lambda$.
     **if** any $(\gamma_i)_{i \in I_T}$ violates the box constraints (11) **then**
       Compute the largest $\rho$ s. t., for all $i \in I_T$ $\gamma_i^{\text{new}} = \gamma_i^{\text{old}} + \rho(\gamma_i - \gamma_i^{\text{old}})$ obey box constraints
       Let $j$ denote the index of $(\gamma_i^{\text{new}})_{i \in I_T}$ at bound,
       $I_T = I_T \setminus \{j\}, \overline{I_T} = \overline{I_T} \cup \{j\}$
       $\gamma_j^{\text{old}} = \gamma_j^{\text{new}}$
     **else**
       **for all** $i \in I_T$ **do** $\gamma_i^{\text{new}} = \gamma_i$
       **if** any $(y_i(f(\mathbf{x}_i) + b))_{i \in \overline{I_T}}$ violates primal constraints (10) **then**
         select $i$ with violated constraint
         $\overline{I_T} = \overline{I_T} \setminus \{i\}, I_T = I_T \cup \{i\}$
       **else**
         exact convergence
       **end if**
       **for all** $i \in I_T$ **do** $\gamma_i^{\text{old}} = \gamma_i^{\text{new}}$
     **end if**
  **until** convergence
**output**  $f, b$.

---

The exact convergence is obtained when all constraints are fulfilled, that is, when all examples belong to the same subset at the begining and the end of the main loop. However, it is possible to relax the convergence criteria while having a good control on the precision on the solution by monitoring the duality gap, that is the difference between the primal and the dual objectives, respectively provided in the definition of Problems (10) and (11).

Table 1: Performances in terms of average test loss, rejection rate and misclassification rate (rejection is not an error) with $r_+ = r_- = 0.45$, for the three rejection methods over four different datasets.

| | | Average Test Loss | Rejection rate (%) | Error rate (%) |
|---|---|---|---|---|
| Wdbc | Naive | $2.9 \pm 1.6$ | 0.7 | 2.6 |
| | B&W's | $3.5 \pm 1.8$ | 3.9 | 1.8 |
| | Our's | $2.9 \pm 1.7$ | 1.2 | 2.4 |
| Liver | Naive | $28.9 \pm 5.4$ | 3.3 | 27.4 |
| | B&W's | $30.9 \pm 4.0$ | 34.5 | 15.4 |
| | Our's | $28.8 \pm 5.1$ | 7.9 | 25.2 |
| Thyroid | Naive | $4.1 \pm 2.9$ | 0.9 | 3.7 |
| | B&W's | $4.4 \pm 2.7$ | 6.1 | 1.6 |
| | Our's | $3.7 \pm 2.7$ | 2.1 | 2.8 |
| Pima | Naive | $23.7 \pm 1.9$ | 7.5 | 20.3 |
| | B&W's | $24.7 \pm 2.1$ | 24.3 | 13.8 |
| | Our's | $23.1 \pm 1.3$ | 6.9 | 20.0 |

**Theorem 2.** *Algorithm 1 converges in a finite number of steps to the exact solution of (11).*

*Proof.* The proof follows the ones used to prove the convergence of active set methods in general, and SimpleSVM in particular, see Propositon 1 in (Vishwanathan et al., 2003)). □

## 5 Experiments

We compare the performances of three different rejection schemes based on SVMs. For this purpose, we selected the datasets from the UCI repository related to medical problems, as medical decision making is an application domain for which rejection is of primary importance. Since these datasets are small, we repeated 10 trials for each problem. Each trial consists in splitting randomly the examples into a training set with 80 % of examples and an independent test set. Note that the training examples were normalized to zero-mean and unit variance before cross-validation (test sets were of course rescaled accordingly).

In a first series of experiments, to compare our decision rule with the one proposed by Bartlett and Wegkamp (2008) (B&W's), we used symmetric costs: $c_+ = c_- = 1$ and $r_+ = r_- = r$. We also chose $r = 0.45$, which corresponds to rather low rejection rates, in order to favour different behaviors between these two decision rules (recall that they are identical for $r \simeq 0.24$). Besides the double hinge loss, we also implemented a "naive" method that consists in running the standard SVM algorithm (using the hinge loss) and selecting a symmetric rejection region around zero by cross-validation.

For all methods, we used Gaussian kernels. Model selection is performed by cross-validation. This includes the selection of the kernel widths, the regularization parameter $C$ for all methods and additionally of the rejection thresholds for the naive method. Note that B&W's and our decision rules are based on learning with the double-hinge loss. Hence, the results displayed in Table 1 only differ due to the size of the rejection region, and to the disparities that arise from the choice of hyper-parameters that may arise in the cross-validation process (since the decision rules differ, the cross-validation scores differ also).

Table 1 summarizes the averaged performances over the 10 trials. Overall, all methods lead to equivalent average test losses, with an unsignificant but consistent advantage for our decision rule. We also see that the naive method tends to reject fewer test examples than the consistent methods. This means that, for comparable average losses, the decision rules based on the scores learned by minimizing the double hinge loss tend to classify more accurately the examples that are not rejected, as seen on the last column of the table.

For noisy problems such as Liver and Pima, we observed that reducing rejection costs considerably decrease the error rate on classified examples (not shown on the table). The performances of the two learning methods based on the double-hinge get closer, and there is still no significant gain

compared to the naive approach. Note however that the symmetric setting is favourable to the naive approach, since we only have to estimate a single decision thershold. We are experimenting to see whether the double-hinge loss shows more substantial improvements for asymmetric losses and for larger training sets.

# 6   Conclusion

In this paper we proposed a new solution to the general problem of classification with a reject option. The double hinge loss was derived from the simple desiderata to obtain accurate estimates of posterior probabilities only in the vicinity of the decision boundaries. Our formulation handles asymmetric misclassification and rejection costs and compares favorably to the one of Bartlett and Wegkamp (2008).

We showed that for suitable kernels, including usual ones such as the Gaussian kernel, training a kernel machine with the double hinge loss provides a universally consistent classifier with reject option. Furthermore, the loss provides sparse solutions, with a limited number of support vectors, similarly to the standard L1-SVM classifier.

We presented what we believe to be the first principled and efficient implementation of SVMs for classification with a reject option. Our optimization scheme is based on an active set method, whose complexity compares to standard SVMs. The dimension of our quadratic program is bounded by the number of examples, and is effectively limited to the number of support vectors. The only computational overhead is brought by monitoring five categories of examples, instead of the three ones considered in standard SVMs (support vector, support at bound, inactive example).

Our approach for deriving the double hinge loss can be used for other decision problems relying on conditional probabilities at specific values or in a limited range or values. As a first example, one may target the estimation of discretized confidence ratings, such as the ones reported in weather forecasts. Multi-category classification also belongs to this class of problems, since there, decisions rely on having precise conditional probabilities within a predefined interval.

**Acknowledgements**

This work was supported in part by the French national research agency (ANR) through project GD2GS, and by the IST Programme of the European Community through project DIRAC.

# References

Bartlett, P. L., & Tewari, A. (2007). Sparseness vs estimating conditional probabilities: Some asymptotic results. *Journal of Machine Learning Research*, *8*, 775–790.

Bartlett, P. L., & Wegkamp, M. H. (2008). Classification with a reject option using a hinge loss. *Journal of Machine Learning Research*, *9*, 1823–1840.

Chow, C. K. (1970). On optimum recognition error and reject tradeoff. *IEEE Trans. on Info. Theory*, *16*, 41–46.

Fumera, G., & Roli, F. (2002). Support vector machines with embedded reject option. *Pattern Recognition with Support Vector Machines: First International Workshop* (pp. 68–82). Springer.

Grandvalet, Y., Mariéthoz, J., & Bengio, S. (2006). A probabilistic interpretation of SVMs with an application to unbalanced classification. *NIPS 18* (pp. 467–474). MIT Press.

Herbei, R., & Wegkamp, M. H. (2006). Classification with reject option. *The Canadian Journal of Statistics*, *34*, 709–721.

Kwok, J. T. (1999). Moderating the outputs of support vector machine classifiers. *IEEE Trans. on Neural Networks*, *10*, 1018–1031.

Steinwart, I. (2005). Consistency of support vector machine and other regularized kernel classifiers. *IEEE Trans. on Info. Theory*, *51*, 128–142.

Vapnik, V. N. (1995). *The nature of statistical learning theory*. Springer Series in Statistics. Springer.

Vishwanathan, S. V. N., Smola, A., & Murty, N. (2003). SimpleSVM. *Proceedings of the Twentieth International Conference on Machine Learning* (pp. 68–82). AAAI.

